# Neural Network Simulation
# of
# Somatosensory Representational Plasticity

**Kamil A. Grajski**
Ford Aerospace
San Jose, CA 95161-9041
kamil@wdl1.fac.ford.com

**Michael M. Merzenich**
Coleman Laboratories
UC San Francisco
San Francisco, CA 94143

## ABSTRACT

The brain represents the skin surface as a topographic map in the somatosensory cortex. This map has been shown experimentally to be modifiable in a use-dependent fashion throughout life. We present a neural network simulation of the competitive dynamics underlying this cortical plasticity by detailed analysis of receptive field properties of model neurons during simulations of skin co-activation, cortical lesion, digit amputation and nerve section.

## 1  INTRODUCTION

Plasticity of adult somatosensory cortical maps has been demonstrated experimentally in a variety of maps and species (Kass, et al., 1983; Wall, 1988). This report focuses on modelling primary somatosensory cortical plasticity in the adult monkey.

We model the long-term consequences of four specific experiments, taken in pairs. With the first pair, behaviorally controlled stimulation of restricted skin surfaces (Jenkins, et al., 1990) and induced cortical lesions (Jenkins and Merzenich, 1987), we demonstrate that Hebbian-type dynamics is sufficient to account for the inverse relationship between cortical magnification (area of cortical map representing a unit area of skin) and receptive field size (skin surface which when stimulated excites a cortical unit) (Sur, et al., 1980; Grajski and Merzenich, 1990). These results are obtained with several variations of the basic model. We conclude that relying solely on cortical magnification and receptive field size will not disambiguate the contributions of each of the myriad circuits known to occur in the brain. With the second pair, digit amputation (Merzenich, et al., 1984) and peripheral nerve cut (without regeneration) (Merzenich, et al., 1983), we explore the role of local excitatory connections in the model

cortex (Grajski, *submitted* ).

Previous models have focused on the self-organization of topographic maps in general (Willshaw and von der Malsburg, 1976; Takeuchi and Amari, 1979; Kohonen, 1982; among others). Ritter and Schulten (1986) specifically addressed somatosensory plasticity using a variant of Kohonen's self-organizing mapping. Recently, Pearson, et al., (1987), using the framework of the Group Selection Hypothesis, have also modelled aspects of normal and reorganized somatosensory plasticity.

Elements of the present study have been published elsewhere (Grajski and Merzenich, 1990).

## 2  THE MODEL

### 2.1  ARCHITECTURE

The network consists of three heirarchically organized two-dimensional layers shown in Figure 1A.

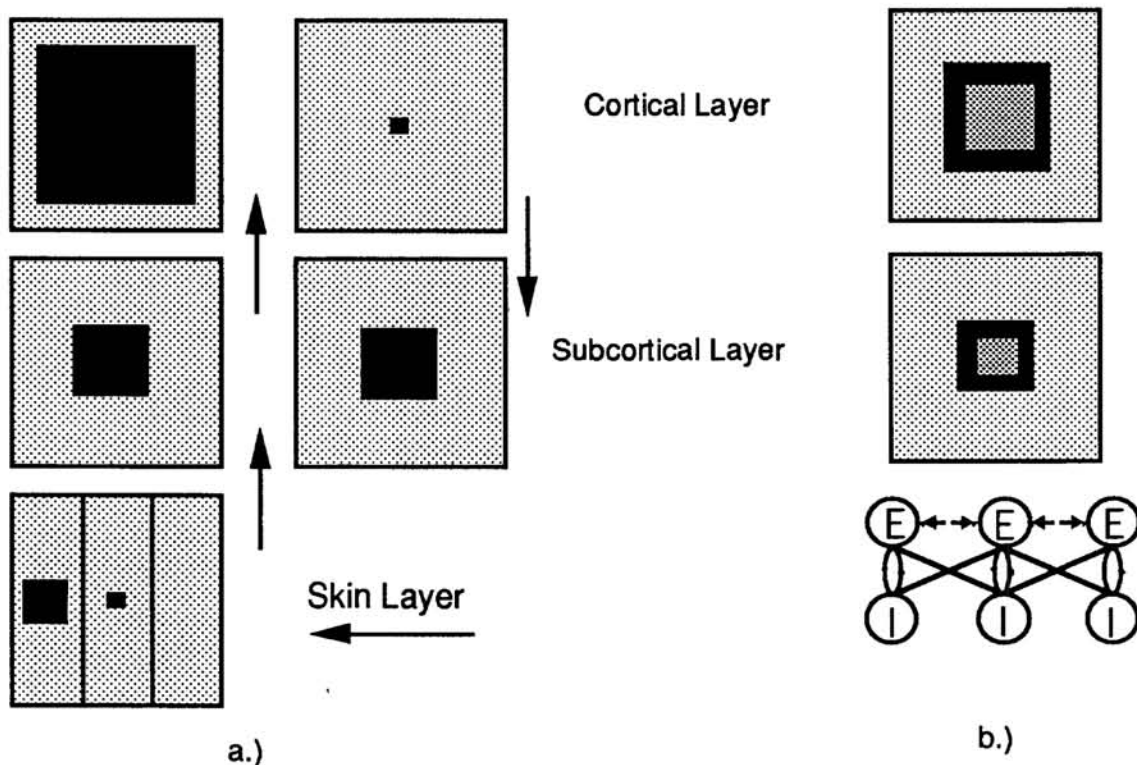

**Figure 1:** Network architecture.

The divergence of projections from a single skin site to subcortex (SC) and its subsequent projection to cortex (C) is shown at left: Skin (S) to SC, 5 x 5; SC to C, 7 x 7. S is "partitioned" into three 15 x 5 "digits" Left, Center and Right. The standard S stimulus used in all simulations is shown lying on digit Left. The projection from C to SC E and I cells is shown at right. Each node in the SC and C layers contains an excitatory (E) and inhibitory cell (I) as shown in Figure 1B. In C, each E cell forms excitatory connections with a 5 x 5 patch of I cells; each I cell forms inhibitory con-

nections with a 7 x 7 path of E cells. In SC, these connections are 3 x 3 and 5 x 5, respectively. In addition, in C only, E cells form excitatory connections with a 5 by 5 patch of E cells. The spatial relationship of E and I cell projections for the central node is shown at left (C E to E shown in light gray, C I to E shown in black).

## 2.2   DYNAMICS

The model neuron is the same for all E and I cells: an RC-time constant membrane which is depolarized and (additively) hyperpolarized by linearly weighted connections:

$$\dot{u}_i^{C,E} = -\tau_m u_i^{C,E} + \sum_j v_j^{SC,E} w_{ij}^{C,E:SC,E} + \sum_j v_j^{C,E} w_{ij}^{C,E:C,E} - \sum_j v_j^{C,J} w_{ij}^{C,E:C,J}$$

$$\dot{u}_i^{C,J} = -\tau_m u_i^{C,J} + \sum_j v_j^{C,E} w_{ij}^{C,J:C,E}$$

$$\dot{u}_i^{SC,E} = -\tau_m u_i^{SC,E} + \sum_j \delta_j^S w_{ij}^{SC,E:S} + \sum_j v_j^{C,E} w_{ij}^{SC,E:C,E} - \sum_j v_j^{SC,E} w_{ij}^{SC,E:SC,J}$$

$$\dot{u}_i^{SC,J} = -\tau_m u_i^{SC,J} + \sum_j v_j^{SC,E} w_{ij}^{SC,J:SC,E} + \sum_j v_j^{C,E} w_{ij}^{SC,J:C,E} - \sum_j v_j^{SC,E} w_{ij}^{SC,E:SC,J}$$

$u_i^{X,Y}$ - membrane potential for unit i of type Y on layer X; $v_i^{X,Y}$ - firing rate for unit i of type Y on layer X; $\delta_j^S$ - skin units are OFF (=0) or ON (=1); $\tau_m$ - membrane time constant (with respect to unit time); $w_{ij}^{post(x,y):pre(X,Y)}$ - connection to unit i of post-synaptic type y on postsynaptic layer x from units of presynaptic type Y on presynaptic layer X. Each summation term is normalized by the number of incoming connections (corrected for planar boundary conditions) contributing to the term. Each unit converts membrane potential to a continuous-valued output value $v_i$ via a sigmoidal function representing an average firing rate ($\beta = 4.0$):

$$v_i = g(u_i) = \begin{cases} \frac{1}{2}(1+\tanh(\beta(u_i-\frac{1}{2}))) & u_i \geq 0.02, \\ 0 & u_i < 0.02 \end{cases}$$

## 2.3   SYNAPTIC PLASTICITY

Synaptic strength is modified in three ways: a.) activity-dependent change; b.) passive decay; and c.) normalization. Activity-dependent and passive decay terms are as follows:

$$\Delta w_{ij} = -\tau_{syn} w_{ij} + \alpha v_i v_j$$

$w_{ij}$ - connection from cell j to cell i; $\tau_{syn}=0.01\tau_m=0.005$ - time constant for passive synaptic decay; $\alpha=0.05$, the maximum activity-dependent step change; $v_j, v_i$ - pre- and post-synaptic output values, respectively. Further modification occurs by a multiplicative normalization performed over the incoming connections for each cell. The normalization is such that the summed total strength of incoming connections is R:

$$\frac{1}{N_i} \Sigma_j w_{ij} = R$$

$N_i$ - number of incoming connections for cell i; $w_{ij}$ - connection from cell j to cell i; $R = 2.0$ - the total resource available to cell i for redistribution over its incoming connections.

## 2.4 MEASURING CORTICAL MAGNIFICATION, RECEPTIVE FIELD AREA

Cortical magnification is measured by "mapping" the network, e.g., noting which 3x3 skin patch most strongly drives each cortical E cell. The number of cortical nodes driven maximally by the same skin site is the cortical magnification for that skin site. Receptive field size for a C (SC) layer E cell is estimated by stimulating all possible 3x3 skin patches (169) and noting the peak response. Receptive field size is defined as the number of 3x3 skin patches which drive the unit at $\geq 50\%$ of its peak response.

# 3  SIMULATIONS

## 3.1 FORMATION OF THE TOPOGRAPHIC MAP ENTAILS REFINEMENT OF SYNAPTIC PATTERNING

The location of individual connections is fixed by topographic projection; initial strengths are drawn from a Gaussian distribution ($\mu = 2.0$, $\sigma^2 = 0.2$). Standard-sized skin patches are stimulated in random sequence with no double-digit stimulation. (Mapping includes tests for double-digit receptive fields.) For each patch, the network is allowed to reach steady-state while the plasticity rule is ON. Synaptic strengths are then renormalized. Refinement continues until two conditions are met: a.) fewer than 5% of all E cells change their receptive field location; and b.) receptive field areas (using the 50% criterion) change by no more than ±1 unit area for 95% of E cells. (See Figures 2 and 3 in Merzenich and Grajski, 1990; Grajski, *submitted* ).

## 3.2 RESTRICTED SKIN STIMULATION GIVES INCREASED MAGNIFICA-TION, DECREASED RECEPTIVE FIELD SIZE

Jenkins, et al., (1990) describe a behavioral experiment which leads to cortical somatotopic reorganization. Monkeys are trained to maintain contact with a rotating disk situated such that only the tips of one or two of their longest digits are stimulated. Monkeys are required to maintain this contact for a specified period of time in order to receive food reward. Comparison of pre- and post-stimulation maps (or the latter with maps obtained after varying periods without disk stimulation) reveal up to nearly 3-fold differences in cortical magnification and reduction in receptive field size for stimulated skin.

We simulate the above experiment by extending the refinement process described above, but with the probability of stimulating a restricted skin region increased 5:1. (See Grajski and Merzenich (1990), Figure 4.) Figure 2 illustrates the change in size (left) and synaptic patterning (right) for a single representative cortical receptive field.

**Figure 2:** Representative co-activation induced receptive field changes.

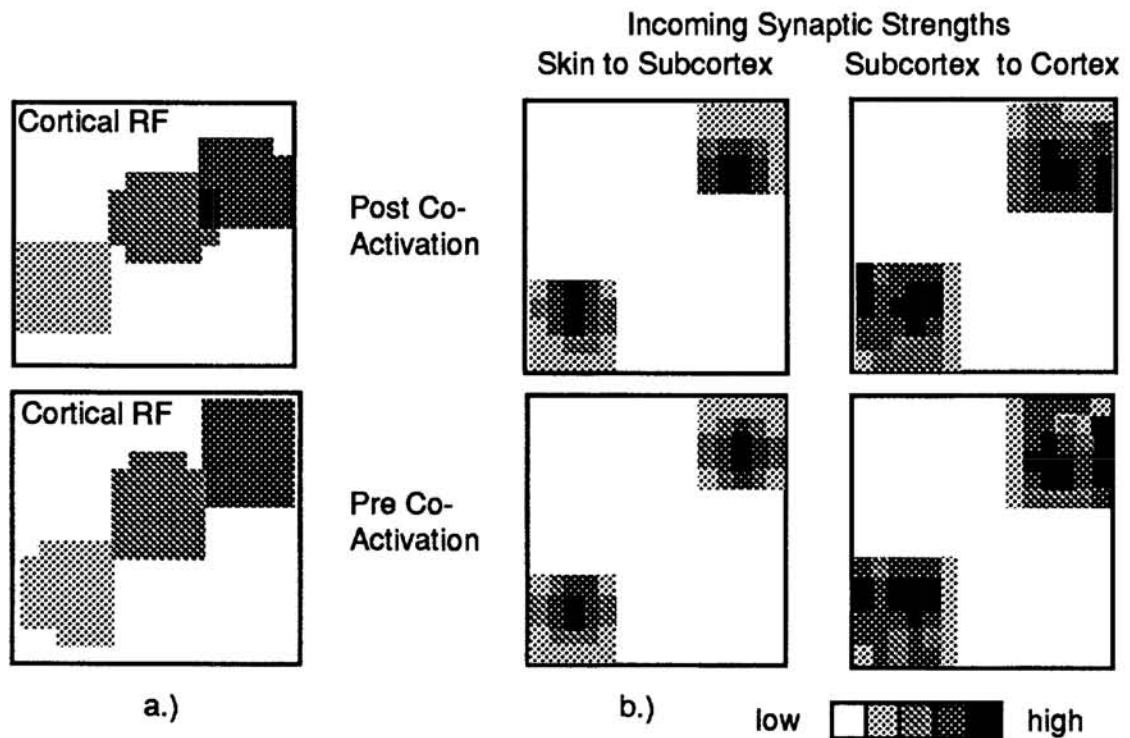

a.)                                                    b.)       low [▢▨▨▨] high

## 3.3   AN INDUCED, FOCAL CORTICAL LESIONS GIVES DECREASED MAGNIFICATION, INCREASED RECEPTIVE FIELD SIZE

The inverse magnification rule predicts that a decrease in cortical magnification is accompanied by an increase in receptive field areas. Jenkins, et al., (1987) confirmed this hypothesis by inducing focal cortical lesions in the representation of restricted hand surfaces, e.g. a single digit. Changes included: a.) a re-emergence of a representation of the skin formerly represented in the now lesioned zone in the intact surrounding cortex; b.) the new representation is at the expense of cortical magnification of skin originally represented in those regions; so that c.) large regions of the map contain neurons with abnormally large receptive fields.

We simulate this experiment by eliminating the incoming and outgoing connections of the cortical layer region representing the middle digit. The refinement process described above is continued under these new conditions until topographic map and receptive field size measures converge. The re-emergence of representation and changes in distributions of receptive field areas are shown in Grajski and Merzenich, (1990) Figure 5. Figure 3 below illustrates the change in size and location of a representative (sub) cortical receptive field.

## 3.4   SEVERAL MODEL VARIANTS REPRODUCE THE INVERSE MAGNIFICATION RULE

Repeating the above simulations using networks with no descending projections or using networks with no descending and no cortical mutual exciatation, yields largely normal topography and co-activation results. Restricting plasticity to excitatory pathways alone also yields qualitatively similar results. (Studies with a two-layer network

yield qualitatively similar results.) Thus, the refinement and co-activation experiments alone are insufficient to discriminate fundamental differences between network variants.

**Figure 3:** Representative cortical lesion induced receptive field changes.

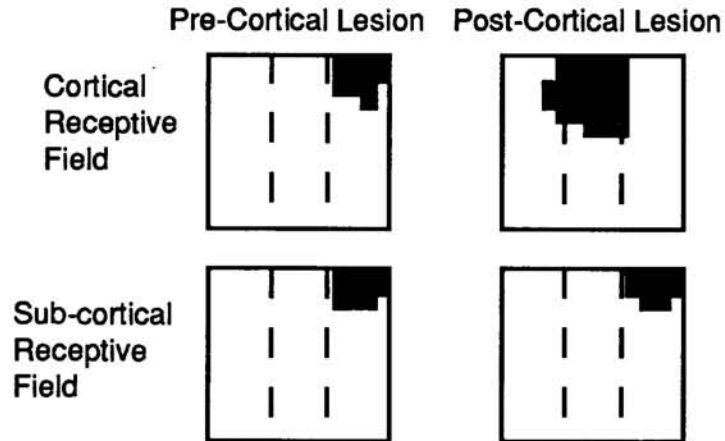

### 3.5 MUTUALLY EXCITATORY LOCAL CORTICAL CONNECTIONS MAY BE CRITICAL FOR SIMULATING EFFECTS OF DIGIT AMPUTATION AND NERVE SECTION

The role of lateral excitation in the cortical layer is made clearer through simulations of nerve section and digit amputation experiments (Merzenich, et al., 1983; Merzenich, et al., 1984; see also Wall, 1988). The feature of interest here is the cortical distance over which reorganization is observed. Following cessation of peripheral input from digit 3, for example, the surrounding representations (digits 2 and 4) expand into the now silenced zone. Not only expansion is observed. Neurons in the surrounding representations up to several 100's of microns distant from the silenced zone shift their receptive fields. The shift is such that the receptive field covers skin sites closer to the silenced skin.

The deafferentation experiment is simulated by eliminating the connection between the skin layer CENTER digit (central 1/3) and SC layers and then proceeding with refinement with the usual convergence checks. Simulations are run for three network architectures. The "full" model is that described above. Two other models strip the descending and both descending and lateral excitatory connections, respectively.

Figure 4 shows features of reorganization: the conversion of initially silenced zones, or refinement of initially large, low amplitude fields to normal-like fields (a-c). Importantly, the receptive field farthest away from the initially silenced representation (d) undergoes a shift towards the deafferented skin. The shift is comprised of a translation in the receptive field peak location as well as an increase (below the 50% amplitude threshold, but increases range 25 - 200%) in the regions surrounding the peak and facing the silenced cortical zone (shown in light shading). Only the "full" model evolves expanded *and* shifted representations. These results are preliminary in that no parameter adjustments are made in the other networks to coax a result. It may simply be a matter of not enough excitation in the other cases. Nevertheless, these results show that local cortical excitation can contribute critical activity for reorganization.

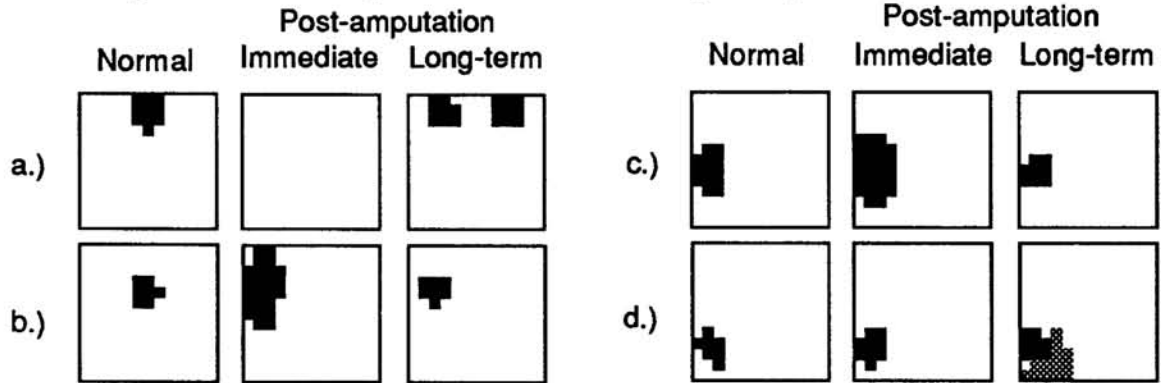

**Figure 4:** Summary of immediate and long-term post-amputation effects.

## 4  CONCLUSION

We have shown that a.) Hebbian-type dynamics is sufficient to account for the quantitative inverse relationship between cortical magnification and receptive field size; and b.) cortical magnification and receptive field size alone are insufficient to distinguish between model variants.

*Are these results just "so much biological detail?"* No. The inverse magnification-receptive field rule applies nearly universally in (sub)cortical topographic maps; it reflects a fundamental principle of brain organization. For instance, experiments revealing the operation of mechanisms possibly similar to those modelled above have been observed in the visual system. Wurtz, et al., (1990) have observed that following chemically induced focal lesions in visual area MT, surviving neurons' visual receptive field area increased. For a review of use-dependent receptive field plasticity in the auditory system see Weinberger, et al., (1990).

Research in computational neuroscience has long drawn on principles of topographic organization. Recent advances include those by Linsker (1989), providing a theoretical (optimization) framework for map formation and those studies linking concepts related to localized receptive fields with adaptive nets (Moody and Darken, 1989; see Barron, this volume). The experimental and modelling issues discussed here offer an opportunity to sustain and further enhance the synergy inherent in this area of computational neuroscience.

### 4.0.1  Acknowldegements

This research supported by NIH grants (to MMM) NS10414 and GM07449, Hearing Research Inc., the Coleman Fund and the San Diego Supercomputer Center. KAG gratefully acknowledges helpful discussions with Terry Allard, Bill Jenkins, John Pearson, Gregg Recanzone and especially Ken Miller.

### 4.0.2  References

Grajski, K. A. and M. M. Merzenich. (1990). Hebb-type dynamics is sufficient to account for the inverse magnification rule in cortical somatotopy. *In Press. Neural Computation.* Vol. 2. No. 1.

Jenkins, W. M. and M. M. Merzenich. (1987). Reorganization of neocortical representations after brain injury. In: **Progress in Brain Research.** Vol. 71. Seil, F. J., et al., Eds. Elsevier. pgs. 249-266.

Jenkins, W. M., et al., (1990). Functional reorganization of primary somatosensory cortex in adult owl monkeys after behaviorally controlled tactile stimulation. *J. Neurophys. In Press.*

Kaas, J. H., M. M. Merzenich and H. P. Killackey. (1983). The reorganization of somatosensory cortex following peripheral nerve damage in adult and developing mammals. *Ann. Rev. Neursci.* 6:325-356.

Kohonen, T. (1982). Self-organized formation of topologically correct feature maps. *Biol. Cyb.* 43:59-69.

Linsker, R. (1989). How to generate ordered maps by maximizing the mutual information between input and output signals. *IBM Research Report No. RC 14624*

Merzenich, M. M., J. H. Kaas, J. T. Wall, R. J. Nelson, M. Sur and D. J. Felleman. (1983). Topographic reorganization of somatosensory cortical areas 3b and 1 in adult monkeys following restricted deafferentation. *Neuroscience.* 8:1:33-55.

Merzenich, M. M., R. J. Nelson, M. P. Stryker, M. Cynader, J. M Zook and A. Schoppman. (1984). Somatosensory cortical map changes following digit amputation in adult monkeys. *J. Comp. Neurol.* 244:591-605.

Moody, J. and C. J. Darken. (1989). Fast learning in networks of locally-tuned processing units. *Neural Computation* 1:281-294.

Pearson, J. C., L. H. Finkel and G. M. Edelman. (1987). Plasticity in the organization of adult cerebral cortical maps. *J. Neurosci.* 7:4209-4223.

Ritter, H. and K. Schulten. (1986). On the stationary state of Kohonen's self-organizing sensory mapping. *Biol. Cyb.* 54:99-106.

Sur, M., M. M. Merzenich and J. H. Kaas. (1980). Magnification, receptive-field area and "hypercolumn" size in areas 3b and 1 of somatosensory cortex in owl monkeys. *J. Neurophys.* 44:295-311.

Takeuchi, A. and S. Amari. (1979). Formation of topographic maps and columnar microstructures in nerve fields. *Biol. Cyb.* 35:63-72.

Wall, J. T. (1988). Variable organization in cortical maps of the skin as an indication of the lifelong adaptive capacities of circuits in the mammalian brain. *Trends in Neurosci.* 11:12:549-557.

Weinberger, N. M., et al., (1990). Retuning auditory cortex by learning: A preliminary model of receptive field plasticity. *Concepts in Neuroscience. In Press.*

Willshaw, D. J. and C. von der Malsburg. (1976). How patterned neural connections can be set up by self-organization. *Proc. R. Soc. Lond.* B. 194:431-445.

Wurtz, R., et al. (1990). Motion to movement: Cerebral cortical visual processing for pursuit eye movements. In: **Signal and sense: Local and global order in perceptual maps.** Gall, E. W., Ed. Wiley: New York. *In Press.*